# Learning multi-class dynamics

**A. Blake, B. North and M. Isard**
Department of Engineering Science, University of Oxford, Oxford OX1 3PJ, UK.
Web: http://www.robots.ox.ac.uk/~vdg/

## Abstract

Standard techniques (eg. Yule-Walker) are available for learning Auto-Regressive process models of simple, directly observable, dynamical processes. When sensor noise means that dynamics are observed only approximately, learning can still been achieved via Expectation-Maximisation (EM) together with Kalman Filtering. However, this does not handle more complex dynamics, involving multiple classes of motion. For that problem, we show here how EM can be combined with the CONDENSATION algorithm, which is based on propagation of random sample-sets. Experiments have been performed with visually observed juggling, and plausible dynamical models are found to emerge from the learning process.

## 1 Introduction

The paper presents a probabilistic framework for estimation (perception) and classification of complex time-varying signals, represented as temporal streams of states. Automated learning of dynamics is of crucial importance as practical models may be too complex for parameters to be set by hand. The framework is particularly general, in several respects, as follows.

**1. Mixed states:** each state comprises a continuous and a discrete component. The continuous component can be thought of as representing the instantaneous position of some object in a continuum. The discrete state represents the current class of the motion, and acts as a label, selecting the current member from a set of dynamical models.

**2. Multi-dimensionality:** the continuous component of a state is, in general, allowed to be multi-dimensional. This could represent motion in a higher dimensional continuum, for example, two-dimensional translation as in figure 1. Other examples include multi-spectral acoustic or image signals, or multi-channel sensors such as an electro-encephalograph.

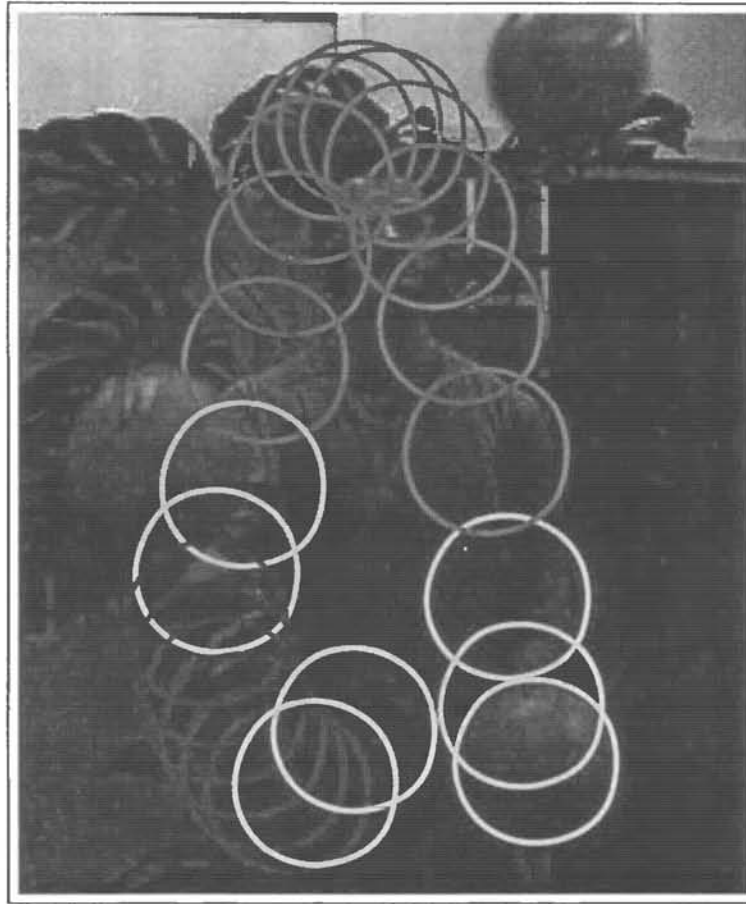

Figure 1: **Learning the dynamics of juggling.** *Three motion classes, emerging from dynamical learning, turn out to correspond accurately to ballistic motion (mid grey), catch/throw (light grey) and carry (dark grey).*

**3. Arbitrary order:** each dynamical system is modelled as an Auto-Regressive Process (ARP) and allowed to have arbitrary order (the number of time-steps of "memory" that it carries.)

**4. Stochastic observations:** the sequence of mixed states is "hidden" — not observable directly, but only via observations, which may be multi-dimensional, and are stochastically related to the continuous component of states. This aspect is essential to represent the inherent variability of response of any real signal sensing system.

Estimation for processes with properties 2,3,4 has been widely discussed both in the control-theory literature as "estimation" and "Kalman filtering" (Gelb, 1974) and in statistics as "forecasting" (Brockwell and Davis, 1996). Learning of models with properties 2,3 is well understood (Gelb, 1974) and once learned can be used to drive pattern classification procedures, as in Linear Predictive Coding (LPC) in speech analysis (Rabiner and Bing-Hwang, 1993), or in classification of EEG signals (Pardey et al., 1995). When property 4 is added, the learning problem becomes harder (Ljung, 1987) because the training sets are no longer observed directly.

Mixed states (property 1) allow for combining perception with classification. Allowing properties 2,4, but restricted to a 0th order ARP (in breach of property 3), gives

Hidden Markov Models (HMM) (Rabiner and Bing-Hwang, 1993), which have been used effectively for visual classification (Bregler, 1997). Learning HMMs is accomplished by the "Baum-Welch" algorithm, a form of Expectation-Maximisation (EM) (Dempster et al., 1977). Baum-Welch learning has been extended to "graphical-models" of quite general topology (Lauritzen, 1996). In this paper, graph topology is a simple chain-pair as in standard HMMs, and the complexity of the problem lies elsewhere — in the generality of the dynamical model.

Generally then, restoring non-zero order to the ARPs (property 3), there is no exact algorithm for estimation. However the estimation problem can be solved by random sampling algorithms, known variously as bootstrap filters (Gordon et al., 1993), particle filters (Kitagawa, 1996), and CONDENSATION (Blake and Isard, 1997). Here we show how such algorithms can be used, with EM, in dynamical learning theory and experiments (figure 1).

## 2  Multi-class dynamics

Continuous dynamical systems can be specified in terms of a continuous state vector $\mathbf{x}_t \in \mathcal{R}^{N_x}$. In machine vision, for example, $\mathbf{x}_t$ represents the parameters of a time-varying shape at time $t$. Multi-class dynamics are represented by appending to the continuous state vector $\mathbf{x}_t$, a discrete state component $y_t$ to make a "mixed" state

$$\mathbf{X}_t = \begin{pmatrix} \mathbf{x}_t \\ y_t \end{pmatrix},$$

where $y_t \in \mathcal{Y} = \{1, \ldots, N_y\}$ is the discrete component of the state, drawn from a finite set of integer labels. Each discrete state represents a class of motion, for example "stroke", "rest" and "shade" for a hand engaged in drawing.

Corresponding to each state $y_t = y$ there is a dynamical model, taken to be a Markov model of order $K^y$ that specifies $p_i(\mathbf{x}_t | \mathbf{x}_{t-1}, \ldots \mathbf{x}_{t-K^y})$. A linear-Gaussian Markov model of order $K$ is an Auto-Regressive Process (ARP) defined by

$$\mathbf{x}_t = \sum_{k=1}^{K} A_k \mathbf{x}_{t-k} + \mathbf{d} + B\mathbf{w}_t$$

in which each $\mathbf{w}_t$ is a vector of $N_x$ independent random $\mathcal{N}(0,1)$ variables and $\mathbf{w}_t$, $\mathbf{w}_{t'}$ are independent for $t \neq t'$. The dynamical parameters of the model are

- deterministic parameters $A_1, A_2, \ldots, A_K$
- stochastic parameters $B$, which are multipliers for the stochastic process $\mathbf{w}_t$, and determine the "coupling" of noise $\mathbf{w}_t$ into the vector valued process $\mathbf{x}_t$.

For convenience of notation, let

$$A = \begin{pmatrix} A_1 & A_2 & \cdots & A_K \end{pmatrix}.$$

Each state $y \in \mathcal{Y}$ has a set $\{A^y, B^y, d^y\}$ of dynamical parameters, and the goal is to learn these from example trajectories. Note that the stochastic parameter $B^y$ is a first-class part of a dynamical model, representing the degree and the shape of uncertainty in motion, allowing the representation of an entire distribution of possible motions for each state $y$. In addition, and independently, state transitions are governed by the transition matrix for a 1st order Markov chain:

$$P(y_t = y' | y_{t-1} = y) = M_{y,y'}.$$

Observations $\mathbf{z}_t$ are assumed to be conditioned purely on the continuous part $\mathbf{x}$ of the mixed state, independent of $y_t$, and this maintains a healthy separation between the modelling of dynamics and of observations. Observations are also assumed to be independent, both mutually and with respect to the dynamical process. The observation process is defined by specifying, at each time $t$, the conditional density $p(\mathbf{z}_t|\mathbf{x}_t)$ which is taken to be Gaussian in experiments here.

## 3   Maximum Likelihood learning

When observations are exact, maximum likelihood estimates (MLE) for dynamical parameters can be obtained from a training sequence $\mathbf{X}_1^* \ldots \mathbf{X}_T^*$ of mixed states. The well known Yule-Walker formula approximates MLE (Gelb, 1974; Ljung, 1987), but generalisations are needed to allow for short training sets (small $T$), to include stochastic parameters $B$, to allow a non-zero offset $\mathbf{d}$ (this proves essential in experiments later) and to encompass multiple dynamical classes.

The resulting MLE learning rule is as follows.

$$A^y \bar{R}^y = \bar{\mathbf{R}}_0^y, \quad \mathbf{d}^y = \frac{1}{T^y - K^y}(R_0^y - A^y \mathbf{R}^y), \quad C^y = \frac{1}{T^y - K^y}\left(\bar{R}_{0,0}^y - A^y(\bar{\mathbf{R}}_0^y)^\top\right),$$

where (omitting the $y$ superscripts for clarity) $C = BB^\top$ and

$$\bar{R} = \begin{pmatrix} \bar{R}_{1,1} & \cdots & \bar{R}_{1,K} \\ \vdots & \ddots & \vdots \\ \bar{R}_{K,1} & \cdots & \bar{R}_{K,K} \end{pmatrix}, \quad \bar{\mathbf{R}}_0 = \begin{pmatrix} \bar{R}_{0,1} & \cdots & \bar{R}_{0,K} \end{pmatrix}, \quad \mathbf{R} = \begin{pmatrix} R_1 \\ \vdots \\ R_K \end{pmatrix},$$

and the first-order moments $R_i$ and (offset-invariant) autocorrelations $\bar{R}_{i,j}$, for each class $y$, are given by

$$R_i^y = \sum_{y_t^* = y} \mathbf{x}_{t-i}^* \quad \text{and} \quad \bar{R}_{i,j}^y = R_{i,j}^y - \frac{1}{T_y - K} R_i^y R_j^{y\top},$$

where

$$R_{i,j}^y = \sum_{y_t = y} \mathbf{x}_{t-i}^* \mathbf{x}_{t-j}^{*\top}; \qquad T_y = \sharp\{t : y_t^* = y\} \equiv \sum_{t:y_t = y} 1.$$

The MLE for the transition matrix $M$ is constructed from relative frequencies as:

$$M_{y,y'} = \frac{T_{y,y'}}{\sum_{y' \in \mathcal{Y}} T_{y,y'}} \quad \text{where} \quad T_{y,y'} = \sharp\{t : y_{t-1}^* = y, y_t^* = y'\}.$$

## 4   Learning with stochastic observations

To allow for stochastic observations, direct MLE is no longer possible, but an EM learning algorithm can be formulated. Its M-step is simply the MLE estimate of the previous section. It might be thought that the E-step should consist simply of computing expectations, for instance $\mathcal{E}[\mathbf{x}_t|\mathcal{Z}_1^T]$, (where $\mathcal{Z}_1^t = (\mathbf{z}_1, \ldots, \mathbf{z}_t)$ denotes a sequence of observations) and treating them as training values $\mathbf{x}_t^*$. This would be incorrect however because the log-likelihood function $\mathcal{L}$ for the problem is not linear in the $\mathbf{x}_t^*$ but quadratic. Instead, we need expectations

$$\mathcal{E}[R_i|\mathcal{Z}_1^T], \quad \mathcal{E}[R_{i,j}|\mathcal{Z}_1^T], \quad \mathcal{E}[T_i|\mathcal{Z}_1^T], \quad \mathcal{E}[T_{i,j}|\mathcal{Z}_1^T],$$

conditioned on the entire training set $\mathcal{Z}_1^T$ of observations, given that $\mathcal{L}$ is linear in the $R_i$, $R_{i,j}$ etc. (Shumway and Stoffer, 1982). These expected values of auto-correlations and frequencies are to be used in place of actual autocorrelations and frequencies in the learning formulae of section 3. The question is, how to compute them. In the special case $\mathcal{Y} = \{1\}$ of single-class dynamics, and assuming a Gaussian observation density, exact methods are available for computing expected moments, using Kalman and smoothing filters (Gelb, 1974), in an "augmented state" filter (North and Blake, 1998). For multi-class dynamics, exact computation is infeasible, but good approximations can be achieved based on propagation of sample sets, using CONDENSATION.

**Forward sampling with backward chaining**

For the purposes of learning, an extended and generalised form of the CONDENSATION algorithm is required. The generalisations allow for mixed states, arbitrary order for the ARP, and backward-chaining of samples. In backward chaining, sample-sets for successive times are built up and stored together with a complete state history back to time $t = 0$. The extended CONDENSATION algorithm is given in figure 2. Note that the algorithm needs to be initialised. This requires that the $y_0$ and $(\mathbf{X}_{-k|0}^{(n)}, \ k = 0, \ldots, K^{y_0} - 1)$ be drawn from a suitable (joint) prior for the multi-class process. One way to do this is to ensure that the training set starts in a known state and to fix the initial sample-values accordingly. Normally, the choice of prior is not too important as it is dominated by data.

At time $t = T$, when the entire training sequence has been processed, the final sample set is

$$\{(\mathbf{X}_{T|T}^{(n)}, \ldots, \mathbf{X}_{0|T}^{(n)}), \pi_T^{(n)}\}, n = 1, \ldots, N\}$$

represents fairly (in the limit, weakly, as $N \to \infty$) the posterior distribution for the entire state sequence $\mathbf{X}_0, \ldots, \mathbf{X}_T$, conditioned on the entire training set $\mathcal{Z}_1^T$ of observations. The expectations of the autocorrelation and frequency measures required for learning can be estimated from the sample set, for example:

$$\mathcal{E}[R_{i,j}^y] \approx \sum_{n=1}^N \pi_T^{(n)} \sum_{\{t: \, y_{t|T}^{(n)} = y\}} \mathbf{x}_{t-i|T}^{(n)} \left( \mathbf{x}_{t-j|T}^{(n)} \right)^\top .$$

An alternative algorithm is a sample-set version of forward-backward propagation (Kitagawa, 1996). Experiments have suggested that probability densities generated by this form of smoothing converge far more quickly with respect to sample set size $N$, but at the expense of computational complexity — $O(N^2)$ as opposed to $O(N \log N)$ for the algorithm above.

## 5 Practical applications

Experiments are reported briefly here on learning the dynamics of juggling using the EM-Condensation algorithm, as in figure 1. An offset $\mathbf{d}^y$ is learned for each class in $\mathcal{Y} = \{1, 2, 3\}$; other dynamical parameters are fixed such that that learning $\mathbf{d}^y$ amounts to learning mean accelerations $\mathbf{a}^y$ for each class. The transition matrix is also learned. From a more or less neutral starting point, learned structure emerges as in figure 3. Around 60 iterations of EM suffice, with $N = 2048$, to learn dynamics in this case. It is clear from the figure that the learned structure is an altogether plausible model for the juggling process.

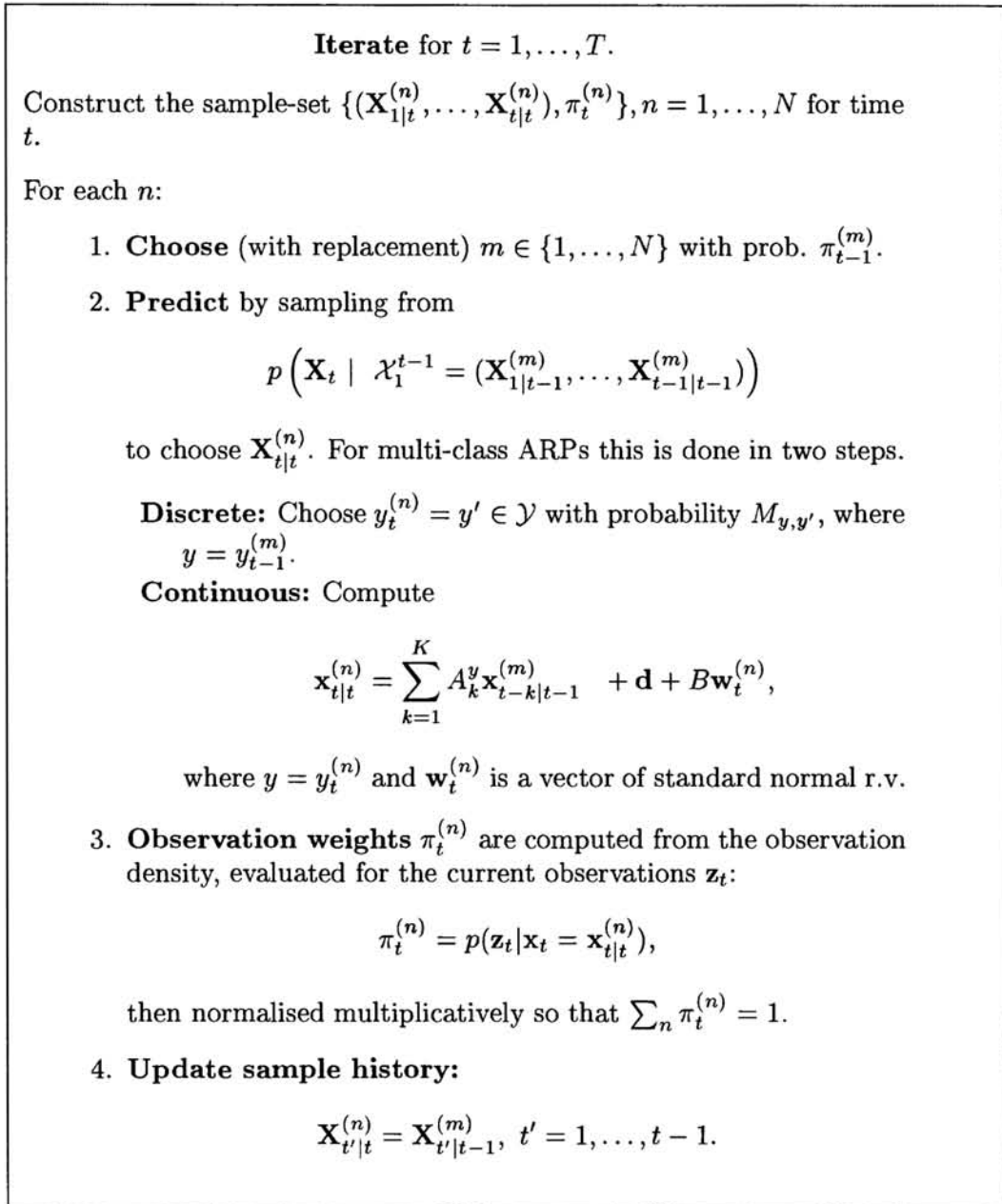

Figure 2: **The CONDENSATION algorithm for forward propagation with backward chaining.**

## Acknowledgements

We are grateful for the support of the EPSRC (AB,BN) and Magdalen College Oxford (MI).

## References

Blake, A. and Isard, M. (1997). The Condensation algorithm — conditional density propagation and applications to visual tracking. In *Advances in Neural Information Processing Systems 9*, pages 361–368. MIT Press.

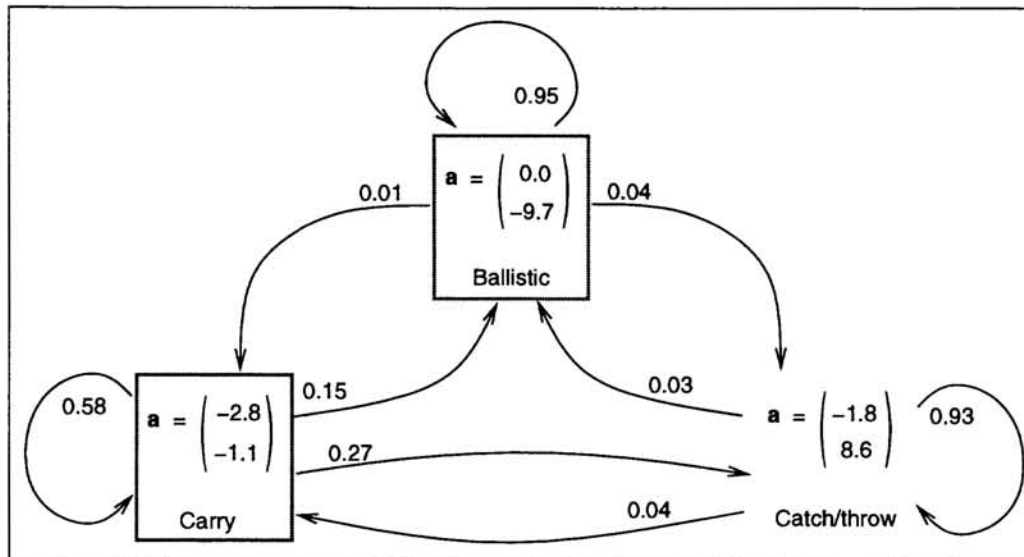

Figure 3: **Learned dynamical model for juggling.** *The three motion classes allowed in this experiment organise themselves into: ballistic motion (acceleration* $\mathbf{a} \approx -g$*); catch/throw; carry. As expected, life-time in the ballistic state is longest, the transition probability of 0.95 corresponding to 20 time-steps or about 0.7 seconds. Transitions tend to be directed, as expected; for example ballistic motion is more likely to be followed by a catch/throw (p = 0.04) than by a carry (p = 0.01). (Acceleration* $\mathbf{a}$ *shown here in units of* $\mathrm{m/s^2}$*.)*

Bregler, C. (1997). Learning and recognising human dynamics in video sequences. In *Proc. Conf. Computer Vision and Pattern Recognition*.

Brockwell, P. and Davis, R. (1996). *Introduction to time-series and forecasting*. Springer-Verlag.

Dempster, A., Laird, M., and Rubin, D. (1977). Maximum likelihood from incomplete data via the EM algorithm. *J. Roy. Stat. Soc. B.*, 39:1–38.

Gelb, A., editor (1974). *Applied Optimal Estimation*. MIT Press, Cambridge, MA.

Gordon, N., Salmond, D., and Smith, A. (1993). Novel approach to nonlinear/non-Gaussian Bayesian state estimation. *IEE Proc. F*, 140(2):107–113.

Kitagawa, G. (1996). Monte Carlo filter and smoother for non-Gaussian nonlinear state space models. *Journal of Computational and Graphical Statistics*, 5(1):1–25.

Lauritzen, S. (1996). *Graphical models*. Oxford.

Ljung, L. (1987). *System identification: theory for the user*. Prentice-Hall.

North, B. and Blake, A. (1998). Learning dynamical models using expectation-maximisation. In *Proc. 6th Int. Conf. on Computer Vision*, pages 384–389.

Pardey, J., Roberts, S., and Tarassenko, L. (1995). A review of parametric modelling techniques for EEG analysis. *Medical Engineering Physics*, 18(1):2–11.

Rabiner, L. and Bing-Hwang, J. (1993). *Fundamentals of speech recognition*. Prentice-Hall.

Shumway, R. and Stoffer, D. (1982). An approach to time series smoothing and forecasting using the EM algorithm. *J. Time Series Analysis*, 3:253–226.
